# Learning to Traverse Image Manifolds

**Piotr Dollár, Vincent Rabaud and Serge Belongie**
University of California, San Diego
{pdollar,vrabaud,sjb}@cs.ucsd.edu

## Abstract

We present a new algorithm, Locally Smooth Manifold Learning (LSML), that learns a warping function from a point on an manifold to its neighbors. Important characteristics of LSML include the ability to recover the structure of the manifold in sparsely populated regions and beyond the support of the provided data. Applications of our proposed technique include embedding with a natural out-of-sample extension and tasks such as tangent distance estimation, frame rate up-conversion, video compression and motion transfer.

## 1   Introduction

A number of techniques have been developed for dealing with high dimensional data sets that fall on or near a smooth low dimensional nonlinear manifold. Such data sets arise whenever the number of modes of variability of the data are much fewer than the dimension of the input space, as is the case for image sequences. Unsupervised manifold learning refers to the problem of recovering the structure of a manifold from a set of unordered sample points. Manifold learning is often equated with dimensionality reduction, where the goal is to find an embedding or 'unrolling' of the manifold into a lower dimensional space such that certain relationships between points are preserved. Such embeddings are typically used for visualization, with the projected dimension being 2 or 3.

Image manifolds have also been studied in the context of measuring distance between images undergoing known transformations. For example, the tangent distance [20, 21] between two images is computed by generating local approximations of a manifold from known transformations and then computing the distance between these approximated manifolds. In this work, we seek to frame the problem of recovering the structure of a manifold as that of directly learning the transformations a point on a manifold may undergo. Our approach, *Locally Smooth Manifold Learning* (LSML), attempts to learn a warping function $\mathcal{W}$ with $d$ degrees of freedom that can take any point on the manifold and generate its neighbors. LSML recovers a first order approximation of $\mathcal{W}$, and by making smoothness assumptions on $\mathcal{W}$ can generalize to unseen points.

We show that LSML can recover the structure of the manifold where data is given, and also in regions where it is not, including regions beyond the support of the original data. We propose a number of uses for the recovered warping function $\mathcal{W}$, including embedding with a natural out-of-sample extension, and in the image domain discuss how it can be used for tasks such as computation of tangent distance, image sequence interpolation, compression, and motion transfer. We also show examples where LSML is used to simultaneously learn the structure of multiple "parallel" manifolds, and even generalize to data on new manifolds. Finally, we show that by exploiting the manifold smoothness, LSML is robust under conditions where many embedding methods have difficulty.

Related work is presented in Section 2 and the algorithm in Section 3. Experiments on point sets and results on images are shown in Sections 4 and 5, respectively. We conclude in Section 6.

## 2 Related Work

Related work can be divided into two categories. The first is the literature on manifold learning, which serves as the foundation for this work. The second is work in computer vision and computer graphics addressing image warping and generative models for image formation.

A number of classic methods exist for recovering the structure of a manifold. Principal component analysis (PCA) tries to find a linear subspace that best captures the variance of the original data. Traditional methods for nonlinear manifolds include self organizing maps, principal curves, and variants of multi-dimensional scaling (MDS) among others, see [11] for a brief introduction to these techniques. Recently the field has seen a number of interesting developments in nonlinear manifold learning. [19] introduced a kernelized version of (PCA). A number of related embedding methods have also been introduced, representatives include LLE [17], ISOMAP [22], and more recently SDE [24]. Broadly, such methods can be classified as spectral embedding techniques [24]; the embeddings they compute are based on an eigenvector decomposition of an $n \times n$ matrix that represents geometrical relationships of some form between the original $n$ points. Out-of-sample extensions have been proposed [3]. The goal of embedding methods (to find structure preserving embeddings) differs from the goals of LSML (learn to traverse the manifold).

Four methods that we share inspiration with are [6, 13, 2, 16]. [6] employs a novel charting based technique to achieve increased robustness to noise and decreased probability of pathological behavior *vs.* LLE and ISOMAP; we exploit similar ideas in the construction of LSML but differ in motivation and potential applicability. [2] proposed a method to learn the tangent space of a manifold and demonstrated a preliminary illustration of rotating a small bitmap image by about $1°$. Work by [13] is based on the notion of learning a model for class specific variation, the method reduces to computing a linear tangent subspace that models variability of each class. [16] shares one of our goals as it addresses the problem of learning Lie groups, the infinitesimal generators of certain geometric transformations.

In image analysis, the number of dimensions is usually reduced via approaches like PCA [15], epitomic representation [12], or generative models like in the realMOVES system developed by Di Bernardo *et al.* [1]. Sometimes, a precise model of the data, like for faces [4] or eyes [14], is even used to reduce the complexity of the data. Another common approach is simply to have instances of an object in different conditions: [5] start by estimating feature correspondences between a novel input with unknown pose and lighting and a stored labeled example in order to apply an arbitrary warp between pictures. The applications range from video texture synthesis [18] and facial expression extrapolation [8, 23] to face recognition [10] and video rewrite [7].

## 3 Algorithm

Let $D$ be the dimension of the input space, and assume the data lies on a smooth $d$-dimensional manifold ($d \ll D$). For simplicity assume that the manifold is diffeomorphic with a subset of $\mathbb{R}^d$, meaning that it can be endowed with a global coordinate system (this requirement can easily be relaxed) and that there exists a continuous bijective mapping $\mathcal{M}$ that converts coordinates $\mathbf{y} \in \mathbb{R}^d$ to points $\mathbf{x} \in \mathbb{R}^D$ on the manifold. The goal of most dimensionality reduction techniques given a set of data points $\mathbf{x}^i$ is to find an embedding $\mathbf{y}^i = \mathcal{M}^{-1}(\mathbf{x}^i)$ that preserves certain properties of the original data like the distances between all points (classical MDS) or the distances or angles between nearby points (*e.g.* spectral embedding methods).

Instead, we seek to learn a warping function $\mathcal{W}$ that can take a point on the manifold and return any neighboring point on the manifold, capturing all the modes of variation of the data. Let us use $\mathcal{W}(\mathbf{x}, \boldsymbol{\epsilon})$ to denote the warping of $\mathbf{x}$, with $\boldsymbol{\epsilon} \in \mathbb{R}^d$ acting on the degrees of freedom of the warp according to the formula $\mathcal{M}$: $\mathcal{W}(\mathbf{x}, \boldsymbol{\epsilon}) = \mathcal{M}(\mathbf{y} + \boldsymbol{\epsilon})$, where $\mathbf{y} = \mathcal{M}^{-1}(\mathbf{x})$. Taking the first order approximation of the above gives: $\mathcal{W}(\mathbf{x}, \boldsymbol{\epsilon}) \approx \mathbf{x} + \mathcal{H}(\mathbf{x})\boldsymbol{\epsilon}$, where each column $\mathcal{H}_{\cdot k}(\mathbf{x})$ of the matrix $\mathcal{H}(\mathbf{x})$ is the partial derivative of $\mathcal{M}$ w.r.t. $\mathbf{y}_k$: $\mathcal{H}_{\cdot k}(\mathbf{x}) = \partial/\partial \mathbf{y}_k \mathcal{M}(\mathbf{y})$. This approximation is valid given $\boldsymbol{\epsilon}$ small enough, hence we speak of $\mathcal{W}$ being an *infinitesimal* warping function.

We can restate our goal of learning to warp in terms of learning a function $\mathcal{H}_\theta : \mathbb{R}^D \rightarrow \mathbb{R}^{D \times d}$ parameterized by a variable $\theta$. Only data points $\mathbf{x}^i$ sampled from one or several manifolds are given. For each $\mathbf{x}^i$, the set $\mathcal{N}^i$ of neighbors is then computed (*e.g.* using variants of nearest neighbor such

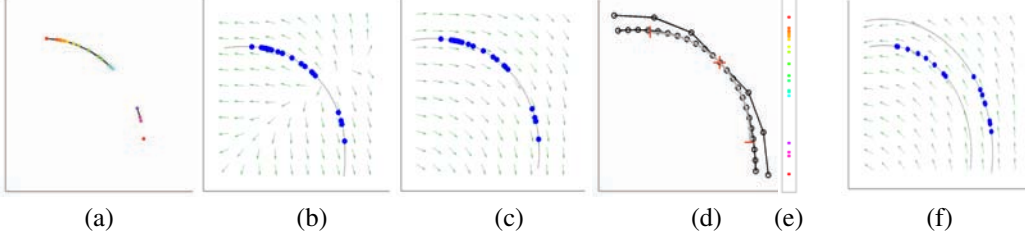

|   (a)   |   (b)   |   (c)   |   (d)   |   (e)   |   (f)   |

Figure 1: **Overview**. Twenty points (n=20) that lie on 1D curve (d=1) in a 2D space (D=2) are shown in (a). Black lines denote neighbors, in this case the neighborhood graph is not connected. We apply LSML to train $\mathcal{H}$ (with $f = 4$ RBFs). $\mathcal{H}$ maps points in $\mathbb{R}^2$ to tangent vectors; in (b) tangent vectors computed over a regularly spaced grid are displayed, with original points (blue) and curve (gray) overlayed. Tangent vectors near original points align with the curve, but note the seam through the middle. Regularization fixes this problem (c), the resulting tangents roughly align to the curve along its entirety. We can traverse the manifold by taking small steps in the direction of the tangent; (d) shows two such paths, generated starting at the red plus and traversing outward in large steps (outer curve) and finer steps (inner curve). This generates a coordinate system for the curve resulting in a 1D embedding shown in (e). In (f) two parallel curves are shown, with n=8 samples each. Training a common $\mathcal{H}$ results in a vector field that more accurately fits each curve than training a separate $\mathcal{H}$ for each (if the structure of the two manifolds was very different this need not be the case).

as $k$NN or $\epsilon$NN), with the constraint that two points can be neighbors only if they come from the same manifold. To proceed, we assume that if $\mathbf{x}^j$ is a neighbor of $\mathbf{x}^i$, there then exists an *unknown* $\epsilon^{ij}$ such that $\mathcal{W}(\mathbf{x}^i, \epsilon^{ij}) = \mathbf{x}^j$ to within a good approximation. Equivalently: $\mathcal{H}_\theta(\mathbf{x}^i)\epsilon^{ij} \approx \mathbf{x}^j - \mathbf{x}^i$. We wish to find the best $\theta$ in the squared error sense (the $\epsilon^{ij}$ being additional free parameters that must be optimized over). The expression of the error we need to minimize is therefore:

$$\text{error}_1(\theta) = \min_{\{\epsilon^{ij}\}} \sum_{i=1}^n \sum_{j \in \mathcal{N}^i} \left\| \mathcal{H}_\theta(\mathbf{x}^i)\epsilon^{ij} - (\mathbf{x}^j - \mathbf{x}^i) \right\|_2^2 \tag{1}$$

Minimizing the above error function can be interpreted as trying to find a warping function that can transform a point into its neighbors. Note, however, that the warping function has only $d$ degrees of freedom while a point may have many more neighbors. This intuition allows us to rewrite the error in an alternate form. Let $\Delta^i$ be the matrix where each column is of the form $(\mathbf{x}^j - \mathbf{x}^i)$ for each neighbor of $\mathbf{x}^i$. Let $\Delta^i = U^i \Sigma^i {V^i}^\top$ be the thin singular value decomposition of $\Delta^i$. Then, one can show [9] that error$_1$ is equivalent to the following:

$$\text{error}_2(\theta) = \min_{\{E^i\}} \sum_{i=1}^n \left\| \mathcal{H}_\theta(\mathbf{x}^i)E^i - U^i\Sigma^i \right\|_F^2 \tag{2}$$

Here, the matrices $E^i$ are the additional free parameters. Minimizing the above can be interpreted as searching for a warping function that directly explains the modes of variation at each point. This form is convenient since we no longer have to keep track of neighbors. Furthermore, if there is no noise and the linearity assumption holds there are at most $d$ non-zero singular values. In practice we use the truncated SVD, keeping at most $2d$ singular values, allowing for significant computational savings.

We now give the remaining details of LSML for the general case [9]. For the case of images, we present an efficient version in Section 5 which uses some basic domain knowledge to avoid solving a large regression. Although potentially any regression technique is applicable, a linear model is particularly easy to work with. Let $\mathbf{f}^i$ be $f$ features computed over $\mathbf{x}^i$. We can then define $\mathcal{H}_\theta(\mathbf{x}^i) = [\Theta^1 \mathbf{f}^i \cdots \Theta^D \mathbf{f}^i]^\top$, where each $\Theta^k$ is a $d \times f$ matrix. Re-arranging error$_2$ gives:

$$\text{error}_{lin}(\theta) = \min_{\{E^i\}} \sum_{i=1}^n \sum_{k=1}^D \left\| {\mathbf{f}^i}^\top {\Theta^k}^\top E^i - U^i_{k\cdot}\Sigma^i \right\|_2^2 \tag{3}$$

Solving simultaneously for $E$ and $\Theta$ is complex, but if either $E$ or $\Theta$ is fixed, solving for the remaining variable becomes a least squares problem (an equation of the form $AXB = C$ can be rewritten as $B^\top \otimes A \cdot \text{vec}(X) = \text{vec}(C)$, where $\otimes$ denotes the Kronecker product and vec the

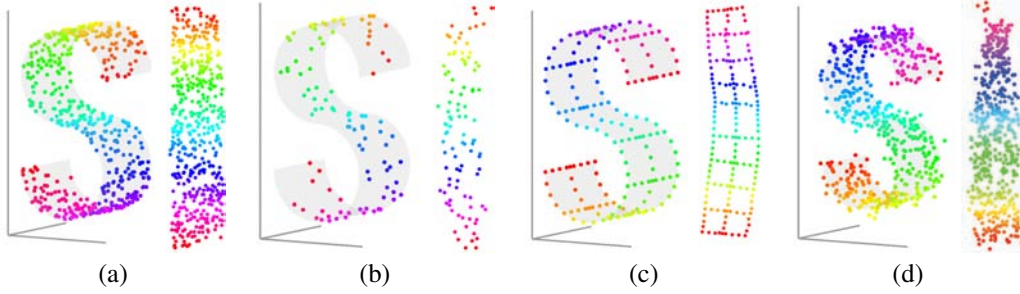

(a)               (b)               (c)               (d)

Figure 2: **Robustness**. LSML used to recover the embedding of the $\mathcal{S}$-curve under a number of sampling conditions. In each plot we show the original points along with the computed embedding (rotated to align vertically), correspondence is indicated by coloring/shading (color was determined by the y-coordinate of the embedding). In each case LSML was run with $f = 8$, $d = 2$, and neighbors computed by $\epsilon$NN with $\epsilon = 1$ (the height of the curve is 4). The embeddings shown were recovered from data that was: (a) densely sampled (n=500) (b) sparsely sampled (n=100), (c) highly structured (n=190), and (d) noisy (n=500, random Gaussian noise with $\sigma = .1$). In each case LSML recovered the correct embedding. For comparison, LLE recovered good embeddings for (a) and (c) and ISOMAP for (a),(b), and (c). The experiments were repeated a number of times yielding similar results. For a discussion see the text.

matrix vectorization function). To solve for $\theta$, we use an alternating minimization procedure. In all experiments in this paper we perform 30 iterations of the above procedure, and while local minima do not seem to be to prevalent, we randomly restart the procedure 5 times. Finally, nowhere in the construction have we enforced that the learned tangent vectors be orthogonal (such a constraint would only be appropriate if the manifold was isometric to a plane). To avoid numerically unstable solutions we regularize the error:

$$\text{error}'_{lin}(\theta) = \text{error}_{lin}(\theta) + \lambda_E \sum_{i=1}^{n} \left\| E^i \right\|_F^2 + \lambda_\theta \sum_{k=1}^{D} \left\| \Theta^k \right\|_F^2 \tag{4}$$

For the features we use radial basis functions (RBFs) [11], the number of basis functions, $f$, being an additional parameter. Each basis function is of the form $f^j(\mathbf{x}) = \exp(-\|\mathbf{x} - \boldsymbol{\mu}^j\|_2^2/2\sigma^2)$ where the centers $\boldsymbol{\mu}^j$ are obtained using K-means clustering on the original data with $f$ clusters and the width parameter $\sigma$ is set to be twice the average of the minimum distance between each cluster and its nearest neighbor center. The feature vectors are then simply defined as $\mathbf{f}^i = [f^1(\mathbf{x}^i) \cdots f^p(\mathbf{x}^i)]^\top$. The parameter $f$ controls the smoothness of the final mapping $\mathcal{H}_\theta$; larger values result in mappings that better fit local variations of the data, but whose generalization abilities to other points on the manifold may be weaker. This is exactly analogous to the standard supervised setting and techniques like cross validation could be used to optimize over $f$.

## 4 Experiments on Point Sets

We begin with a discussion on the intuition behind various aspects of LSML. We then show experiments demonstrating the robustness of the method, followed by a number of applications. In the figures that follow we make use of color/shading to indicate point correspondences, for example when we show the original point set and its embedding.

LSML learns a function $\mathcal{H}$ from points in $\mathbb{R}^D$ to tangent directions that agree, up to a linear combination, with estimated tangent directions at the original training points of the manifold. By constraining $\mathcal{H}$ to be smooth (through use of a limited number of RBFs), we can compute tangents at points not seen during training, including points that may not lie on the underlying manifold. This generalization ability of $\mathcal{H}$ will be central to the types of applications considered. Finally, given multiple non-overlapping manifolds with similar structure, we can train a single $\mathcal{H}$ to correctly predict the tangents of each, allowing information to be shared. Fig. 1 gives a visual tutorial of these different concepts.

LSML appears quite robust. Fig. 2 shows LSML successfully applied for recovering the embedding of the "$\mathcal{S}$-curve" under a number of sampling conditions (similar results were obtained on the "Swiss-roll"). After $\mathcal{H}$ is learned, the embedding is computed by choosing a random point on the manifold

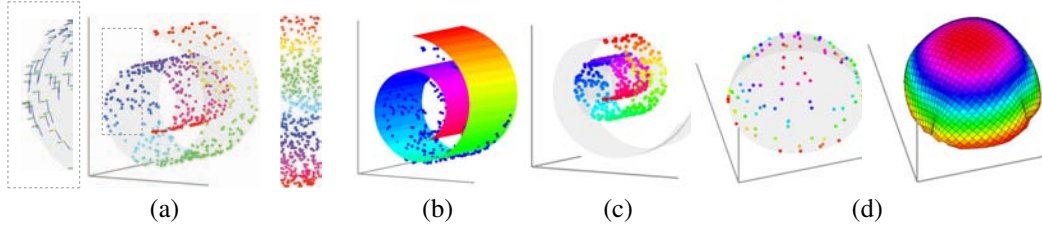

(a)          (b)          (c)          (d)

Figure 3: **Reconstruction**. Reconstruction examples are used to demonstrate quality and generalization of $\mathcal{H}$. (a) Points sampled from the Swiss-roll manifold (middle), some recovered tangent vectors in a zoomed-in region (left) and embedding found by LSML (right). Here $n = 500$ $f = 20$, $d = 2$, and neighbors were computed by $\epsilon$NN with $\epsilon = 4$ (height of roll is 20). Reconstruction of Swiss-roll (b), created by a backprojection from regularly spaced grid points in the embedding (traversal was done from a single original point located at the base of the roll, see text for details). Another reconstruction (c), this time using all points and extending the grid well beyond the support of the original data. The Swiss-roll is extended in a reasonable manner both inward (occluded) and outward. (d) Reconstruction of unit hemisphere (LSML trained with $n = 100$ $f = 6$, $d = 2$, $\epsilon$NN with $\epsilon = .3$) by traversing outward from topmost point, note reconstruction in regions with no points.

and establishing a coordinate system by traversing outward (the same procedure can be used to embed novel points, providing a natural out-of-sample extension). Here we compare only to LLE and ISOMAP using published code. The densely sampled case, Fig. 2(a), is comparatively easy and a number of methods have been shown to successfully recover an embedding. On sparsely sampled data, Fig. 2(b), the problem is more challenging; LLE had problems for $n < 250$ (lowering LLE's regularization parameter helped somewhat). Real data need not be uniformly sampled, see Fig. 2(c). In the presence of noise Fig. 2(d), ISOMAP and LLE performed poorly. A single outlier can distort the shortest path computed by ISOMAP, and LLE does not directly use global information necessary to disambiguate noise. Other methods are known to be robust [6], and in [25] the authors propose a method to "smooth" a manifold as a preprocessing step for manifold learning algorithms; however a full comparison is outside the scope of this work.

Having learned $\mathcal{H}$ and computed an embedding, we can also backproject from a point $\mathbf{y} \in \mathbb{R}^d$ to a point $\mathbf{x}$ on the manifold by first finding the coordinate of the closest point $\mathbf{y}^i$ in the original data, then traversing from $\mathbf{x}^i$ by $\epsilon_j = \mathbf{y}_j - \mathbf{y}_j^i$ along each tangent direction $j$ (see Fig. 1(d)). Fig. 3(a) shows tangents and an embedding recovered by LSML on the Swiss-roll. In Fig. 3(b) we backproject from a grid of points in $\mathbb{R}^2$; by linking adjacent sets of points to form quadrilaterals we can display the resulting backprojected points as a surface. In Fig. 3(c), we likewise do a backprojection (this time keeping all the original points), however we backproject grid points well below and above the support of the original data. Although there is no ground truth here, the resulting extension of the surface seems "natural". Fig. 3(d) shows the reconstruction of a unit hemisphere by traversing outward from the topmost point. There is no isometric mapping (preserving distance) between a hemisphere and a plane, and given a sphere there is actually not even a conformal mapping (preserving angles). In the latter case an embedding is not possible, however, we can still easily recover $\mathcal{H}$ for both (only hemisphere results are shown).

## 5 Results on Images

Before continuing, we consider potential applications of $\mathcal{H}$ in the image domain, including tangent distance estimation, nonlinear interpolation, extrapolation, compression, and motion transfer. We refer to results on point-sets to aid visualization. Tangent distance estimation: $\mathcal{H}$ computes the tangent and can be used directly in invariant recognition schemes such as [21]. Compression: Fig. 3(b,d) suggest how given a reference point and $\mathcal{H}$ nearby points can be reconstructed using $d$ numbers (with distortion increasing with distance). Nonlinear interpolation and extrapolation: points can be generated within and beyond the support of given data (*cf*. Fig. 3); of potential use in tasks such as frame rate up-conversion, reconstructing dropped frames and view synthesis. Motion transfer: for certain classes of manifolds with "parallel" structure (*cf*. Fig. 1(f)), a recovered warp may be used on an entirely novel image. These applications will depend not only on the accuracy of the learned $\mathcal{H}$ but also on how close a set of images is to a smooth manifold.

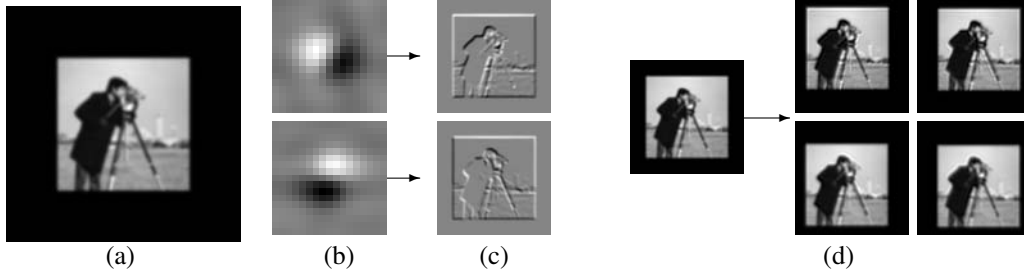

(a)           (b)           (c)           (d)

Figure 4: **The translation manifold**. Here $F^i = X^i$; $s = 17$, $d = 2$ and 9 sets of 6 translated images each were used (not including the cameraman). (a) Zero padded, smoothed test image $\mathbf{x}$. (b) Visualization of learned $\Theta$, see text for details. (c) $\mathcal{H}_\theta(\mathbf{x})$ computed via convolution. (d) Several transformations obtained after multiple steps along manifold for different linear combinations of $\mathcal{H}_\theta(\mathbf{x})$. Some artifacts due to error propagation start to appear in the top figures.

The key insight to working with images is that although images can live in very high dimensional spaces (with $D \approx 10^6$ quite common), we do not have to learn a transformation with that many parameters. Let $\mathbf{x}$ be an image and $\mathcal{H}_{\cdot k}(\mathbf{x})$, $k \in [1, d]$, be the $d$ tangent images. Here we assume that each pixel in $\mathcal{H}_{\cdot k}(\mathbf{x})$ can be computed based only on the information in $s \times s$ patch centered on the corresponding pixel in $\mathbf{x}$. Thus, instead of learning a function $\mathbb{R}^D \rightarrow \mathbb{R}^{D \times d}$ we learn a function $\mathbb{R}^{s^2} \rightarrow \mathbb{R}^d$, and to compute $\mathcal{H}$ we apply the per patch function at each of the $D$ locations in the image. The resulting technique scales independently of $D$, in fact different sized images can be used. The per patch assumption is not always suitable, most notably for transformations that are based only on image coordinate and are independent of appearance.

The approach of Section 3 needs to be slightly modified to accommodate patches. We rewrite each image $\mathbf{x}^i \in \mathbb{R}^D$ as a $s^2 \times D$ matrix $X^i$ where each row contains pixels from one patch in $\mathbf{x}^i$ (in training we sub-sample patches). Patches from all the images are clustered to obtain the $f$ RBFs; each $X^i$ is then transformed to a $f \times D$ matrix $F^i$ that contains the features computed for each patch. The per patch linear model can now be written as $\mathcal{H}_\theta(\mathbf{x}^i) = (\Theta F^i)^\top$, where $\Theta$ is a $d \times f$ matrix (compare with the $D$ $\Theta$s needed without the patch assumption). The error function, which is minimized in a similar way [9], becomes:

$$\text{error}_{img}(\Theta) = \min_{\{E^i\}} \sum_{i=1}^{n} \left\| F^{i^\top} \Theta^\top E^i - U^i \Sigma^i \right\|_F^2 \tag{5}$$

We begin with the illustrative example of translation (Fig. 4). Here, RBFs were not used, instead $F^i = X^i$. The learned $\Theta$ is a $2 \times s^2$ matrix, which can be visualized as two $s \times s$ images as in Fig. 4(b). These resemble derivative of Gaussian filters, which are in fact the infinitesimal generates for translation [16]. Computing the dot product of each column of $\Theta$ with each patch can be done using a convolution. Fig. 4 shows applications of the learned transformations, which resemble translations with some artifacts.

Fig. 5 shows the application of LSML for learning out-of-plane rotation of a teapot. On this size problem training LSML (in MATLAB) takes a few minutes; convergence occurs within about 10 iterations of the minimization procedure. $\mathcal{H}_\theta(\mathbf{x})$ for novel $\mathbf{x}$ can be computed with $f$ convolutions (to compute cross correlation) and is also fast. The outer frames in Fig. 5 highlight a limitation of the approach: with every successive step error is introduced; eventually significant error can accumulate. Here, we used a step size which gives roughly 10 interpolated frames between each pair of original frames. With out-of-plane rotation, information must be created and the problem becomes ambiguous (multiple manifolds can intersect at a single point), hence generalization across images is not expected to be good.

In Fig. 6, results are shown on an eye manifold with 2 degrees of freedom. LSML was trained on sparse data from video of a single eye; $\mathcal{H}_\theta$ was used to synthesize views within and also well outside the support of the original data ($cf$. Fig. 6(c)). In Fig. 6(d), we applied the transformation learned from one person's eye to a single image of another person's eye (taken under the same imaging conditions). LSML was able to start from the novel test image and generate a convincing series of

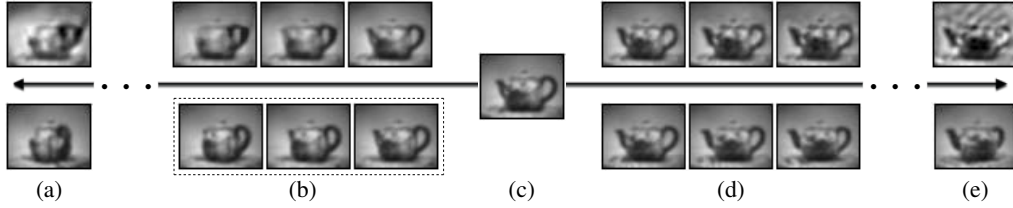

Figure 5: **Manifold generated by out-of-plane rotation of a teapot** (data from [24], sub-sampled and smoothed). Here, $d = 1$, $f = 400$ and roughly 3000 patches of width $s = 13$ were sampled from 30 frames. Bottom row shows the ground truth images; dashed box contains 3 of 30 training images, representing $\sim 8°$ of physical rotation. The top row shows the learned transformation applied to the central image. By observing the tip, handle and the two white blobs on the teapot, and comparing to ground truth data, we can observe the quality of the learned transformation on seen data (b) and unseen data (d), both starting from a single frame (c). The outmost figures (a)(e) shows failure for large rotations.

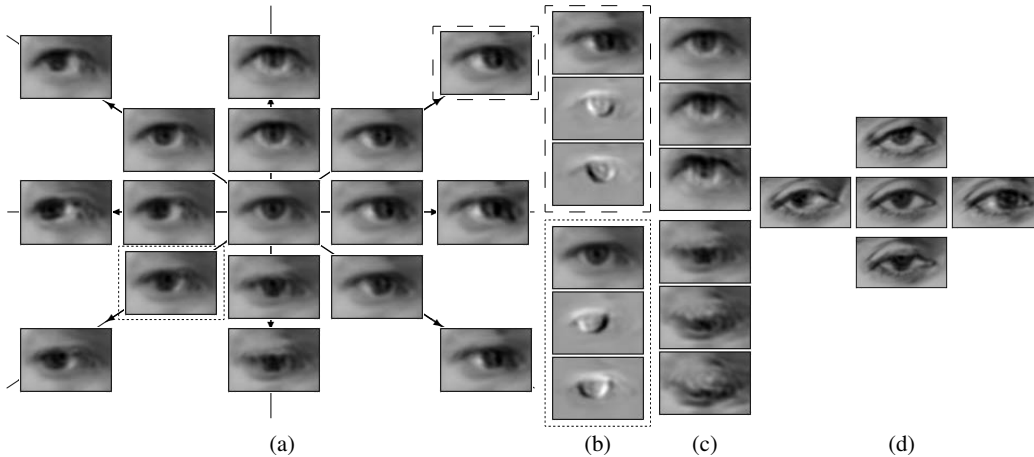

Figure 6: **Traversing the eye manifold**. LSML trained on one eye moving along five different lines (3 vertical and 2 horizontal). Here $d = 2$, $f = 600$, $s = 19$ and around 5000 patches were sampled; 2 frames were considered neighbors if they were adjacent in time. Figure (a) shows images generated from the central image. The inner 8 frames lie just outside the support of the training data (not shown), the outer 8 are extrapolated beyond its support. Figure (b) details $\mathcal{H}_\theta(\mathbf{x})$ for two images in a warping sequence: a linear combination can lead the iris/eyelid to move in different directions (*e.g.* the sum would make the iris go up). Figure (c) shows extrapolation far beyond the training data, *i.e.* an eye wide open and fully closed. Finally, Figure(d) shows how the eye manifold we learned on one eye can be applied on a novel eye not seen during training.

transformations. Thus, motion transfer was possible - $\mathcal{H}_\theta$ trained on one series of images generalized to a different set of images.

# 6 Conclusion

In this work we presented an algorithm, Locally Smooth Manifold Learning, for learning the structure of a manifold. Rather than pose manifold learning as the problem of recovering an embedding, we posed the problem in terms of learning a warping function for traversing the manifold. Smoothness assumptions on $\mathcal{W}$ allowed us to generalize to unseen data. Proposed uses of LSML include tangent distance estimation, frame rate up-conversion, video compression and motion transfer.

We are currently engaged in scaling the implementation to handle large datasets; the goal is to integrate LSML into recognition systems to provide increased invariance to transformations.

## Acknowledgements

This work was funded by the following grants and organizations: NSF Career Grant #0448615, Alfred P. Sloan Research Fellowship, NSF IGERT Grant DGE-0333451, and UCSD Division of Calit2. We would like to thank Sameer Agarwal, Kristin Branson, Matt Tong, and Neel Joshi for valuable input and Anna Shemorry for helping us make it through the deadline.

## References

[1] E. Di Bernardo, L. Goncalves and P. Perona.US Patent 6,552,729: Automatic generation of animation of synthetic characters., 2003.

[2] Y. Bengio and M. Monperrus. Non-local manifold tangent learning. In *NIPS*. 2005.

[3] Y. Bengio, J.F. Paiement, P. Vincent, O. Delalleau, N. Le Roux, and M. Ouimet. Out-of-sample extensions for LLE, isomap, MDS, eigenmaps, and spectral clustering. In *NIPS*, 2004.

[4] D. Beymer and T. Poggio. Face recognition from one example view. In *ICCV*, page 500, Washington, DC, USA, 1995. IEEE Computer Society.

[5] Volker Blanz and Thomas Vetter. Face recognition based on fitting a 3D morphable model. *PAMI*, 25(9):1063–1074, 2003.

[6] M. Brand. Charting a manifold. In *NIPS*, 2003.

[7] Christoph Bregler, Michele Covell, and Malcolm Slaney. Video rewrite: driving visual speech with audio. In *SIGGRAPH*, pages 353–360, 1997.

[8] E. Chuang, H. Deshpande, and C. Bregler. Facial expression space learning. In *Pacific Graphics*, 2002.

[9] P. Dollár, V. Rabaud, and S. Belongie. Learning to traverse image manifolds. Technical Report CS2007-0876, UCSD CSE, Jan. 2007.

[10] G. J. Edwards, T. F. Cootes, and C. J. Taylor. Face recognition using active appearance models. *ECCV*, 1998.

[11] T. Hastie, R. Tibshirani, and J. Friedman. *The Elements of Statistical Learning*. Springer, 2001.

[12] N. Jojic, B. Frey, and A. Kannan. Epitomic analysis of appearance and shape. In *ICCV*, 2003.

[13] D. Keysers, W. Macherey, J. Dahmen, and H. Ney. Learning of variability for invariant statistical pattern recognition. *ECML*, 2001.

[14] T. Moriyama, T. Kanade, J. Xiao, and J. F. Cohn. Meticulously detailed eye region model. *PAMI*, 2006.

[15] H. Murase and S.K. Nayar. Visual learning and recognition of 3D objects from appearance. *IJCV*, 1995.

[16] R. Rao and D. Ruderman. Learning Lie groups for invariant visual perception. In *NIPS*, 1999.

[17] L. K. Saul and S. T. Roweis. Think globally, fit locally: unsupervised learning of low dimensional manifolds. *JMLR*, 2003.

[18] A. Schödl, R. Szeliski, D.H. Salesin, and I. Essa. Video textures. In *SIGGRAPH*, 2000.

[19] B. Schölkopf, A. Smola, and K. Müller. Nonlinear component analysis as a kernel eigenvalue problem. *Neur. Comp.*, 1998.

[20] P. Simard, Y. LeCun, and J. S. Denker. Efficient pattern recognition using a new transformation distance. In *NIPS*, 1993.

[21] P. Simard, Y. LeCun, J. S. Denker, and B. Victorri. Transformation invariance in pattern recognition-tangent distance and tangent propagation. In *Neural Networks: Tricks of the Trade*, 1998.

[22] J. B. Tenenbaum, V. de Silva, and J. C. Langford. A global geometric framework for nonlinear dimensionality reduction. *Science*, 290, 2000.

[23] Joshua B. Tenenbaum and William T. Freeman. Separating style and content with bilinear models. *Neural Computation*, 12(6):1247–1283, 2000.

[24] K. Q. Weinberger and L. K. Saul. Unsupervised learning of image manifolds by semidefinite programming. In *CVPR04*.

[25] Z. Zhang and Zha. Local linear smoothing for nonlinear manifold learning. Technical report, 2003.
